# Estimating the Location and Orientation of Complex, Correlated Neural Activity using MEG

**D.P. Wipf, J.P. Owen, H.T. Attias, K. Sekihara, and S.S. Nagarajan**
Biomagnetic Imaging Laboratory
University of California, San Francisco

## Abstract

The synchronous brain activity measured via MEG (or EEG) can be interpreted as arising from a collection (possibly large) of current dipoles or sources located throughout the cortex. Estimating the number, location, and orientation of these sources remains a challenging task, one that is significantly compounded by the effects of source correlations and the presence of interference from spontaneous brain activity, sensor noise, and other artifacts. This paper derives an empirical Bayesian method for addressing each of these issues in a principled fashion. The resulting algorithm guarantees descent of a cost function uniquely designed to handle unknown orientations and arbitrary correlations. Robust interference suppression is also easily incorporated. In a restricted setting, the proposed method is shown to have theoretically zero bias estimating both the location and orientation of multi-component dipoles even in the presence of correlations, unlike a variety of existing Bayesian localization methods or common signal processing techniques such as beamforming and sLORETA. Empirical results on both simulated and real data sets verify the efficacy of this approach.

## 1 Introduction

Magnetoencephalography (MEG) and related electroencephalography (EEG) use an array of sensors to take electromagnetic field (or voltage potential) measurements from on or near the scalp surface with excellent temporal resolution. In both cases, the observed field is generated by the same synchronous, compact current sources located within the brain. Although useful for research and clinical purposes, accurately determining the spatial distribution of these unknown sources is an open problem. The relevant estimation problem can be posed as follows: The measured electromagnetic signal is $B \in \mathbb{R}^{d_b \times d_t}$, where $d_b$ equals the number of sensors and $d_t$ is the number of time points at which measurements are made. Each unknown source $S_i \in \mathbb{R}^{d_c \times d_t}$ is a $d_c$-dimensional neural current dipole , at $d_t$ timepoints, projecting from the $i$-th (discretized) voxel or candidate location distributed throughout the cortex. These candidate locations can be obtained by segmenting a structural MR scan of a human subject and tesselating the gray matter surface with a set of vertices. $B$ and each $S_i$ are related by the likelihood model

$$B = \sum_{i=1}^{d_s} L_i S_i + \mathcal{E}, \tag{1}$$

where $d_s$ is the number of voxels under consideration, $L_i \in \mathbb{R}^{d_b \times d_c}$ is the so-called lead-field matrix for the $i$-th voxel. The $k$-th column of $L_i$ represents the signal vector that would be observed at the scalp given a unit current source/dipole at the $i$-th vertex with a fixed orientation in the $k$-th direction. It is common to assume $d_c = 2$ (for MEG) or $d_c = 3$ (for EEG), which allows flexible source orientations to be estimated in 2D or 3D space. Multiple methods based on the physical properties of the brain and Maxwell's equations are available for the computation of each $L_i$ [7]. Finally, $\mathcal{E}$ is a noise-plus-interference term where we assume, for simplicity, that columns are drawn independently from $\mathcal{N}(0, \Sigma_\epsilon)$. However, temporal correlations can easily be incorporated if desired using a simple transformation outlined in [3].

To obtain reasonable spatial resolution, the number of candidate source locations will necessarily be much larger than the number of sensors ($d_s \gg d_b$). The salient inverse problem then becomes the ill-posed estimation of regions with significant brain activity, which are reflected by voxels $i$ such that $\|S_i\| > 0$; we refer to these as *active* dipoles or sources. Because the inverse model is severely underdetermined (the mapping from source activity configuration $S \triangleq [S_1, \ldots, S_{d_s}]^T$ to sensor measurement $B$ is many to one), all efforts at source reconstruction are heavily dependent on prior assumptions, which in a Bayesian framework are embedded in the distribution $p(S)$. Such a prior is often considered to be fixed and known, as in the case of minimum current estimation (MCE) [10], minimum variance adaptive beamforming (MVAB) [9], and sLORETA [5]. Alternatively, a number of empirical Bayesian approaches have been proposed that attempt a form of model selection by using the data, whether implicitly or explicitly, to guide the search for an appropriate prior. Examples include variational Bayesian methods and hierarchical covariance component models [3, 6, 8, 12, 13]. While advantageous in many respects, all of these methods retain substantial weaknesses estimating complex, correlated source configurations with unknown orientation in the presence of background interference (e.g., spontaneous brain activity, sensor noise, etc.).

There are two types of correlations that can potentially disrupt the source localization process. First, there are correlations *within* dipole components (meaning the individual rows of $S_i$ are correlated), which always exists to a high degree in real data with unknown orientation (i.e., $d_c > 1$). Secondly, there are correlations *between* different dipoles that are simultaneously active (meaning rows of $S_i$ are correlated with rows of $S_j$ for some voxels $i \neq j$). These correlations are more application specific and may or may not exist. The larger the number of active sources, the greater the chance that both types or correlation can disrupt the estimation process. This issue can be problematic for two reasons. First, failure to accurately account for unknown orientations or correlations can severely disrupt the localization process, leading to a very misleading impression of which brain areas are active. Secondly, the orientations and correlations themselves may have clinical significance.

In this paper, we present an alternative empirical Bayesian scheme that attempts to improve upon existing methods in terms of source reconstruction accuracy and/or computational robustness and efficiency. Section 2 presents the basic generative model which underlies the proposed method and describes the associated inference problem. Section 3 derives a robust algorithm for estimating the sources using this model and proves that each iteration is guaranteed to reduce the associated cost function. It also describes how interference suppression can be naturally incorporated. Section 4 then provides a theoretical analysis of the bias involved in estimating both the location and orientation of active sources, demonstrating that the proposed method has substantial advantages over existing approaches. Finally, Section 5 contains experimental results using our algorithm on both simulated and real data, followed by a brief discussion in Section 6.

## 2  Modeling Assumptions

To begin we invoke the noise model from (1), which fully defines the assumed likelihood

$$p(B|S) \propto \exp\left(-\frac{1}{2}\left\|B - \sum_{i=1}^{d_s} L_i S_i\right\|_{\Sigma_\epsilon^{-1}}^2\right), \tag{2}$$

where $\|X\|_W$ denotes the weighted matrix norm $\sqrt{\mathrm{trace}[X^T W X]}$. The unknown noise covariance $\Sigma_\epsilon$ will be estimated from the data using a variational Bayesian factor analysis (VBFA) model as discussed in Section 3.2 below; for now we will consider that it is fixed and known. Next we adopt the following source prior for $S$:

$$p(S|\Gamma) \propto \exp\left(-\frac{1}{2}\mathrm{trace}\left[\sum_{i=1}^{d_s} S_i^T \Gamma_i^{-1} S_i\right]\right). \tag{3}$$

This is equivalent to applying independently, at each time point, a zero-mean Gaussian distribution with covariance $\Gamma_i$ to each source $S_i$. We define $\Gamma$ to be the $d_s d_c \times d_s d_c$ block-diagonal matrix formed by ordering each $\Gamma_i$ along the diagonal of an otherwise zero-valued matrix. This implies, equivalently, that $p(S|\Gamma) \propto \exp\left(-\frac{1}{2}\mathrm{trace}\left[S^T \Gamma^{-1} S\right]\right)$.

If $\Gamma$ were somehow known, then the conditional distribution $p(S|B, \Gamma) \propto p(B|S)p(S|\Gamma)$ is a fully specified Gaussian distribution with mean and covariance given by

$$\mathrm{E}_{p(S|B,\Gamma)}[S] = \Gamma L^T \left(\Sigma_\epsilon + L\Gamma L^T\right)^{-1} B \tag{4}$$

$$\mathrm{Cov}_{p(\boldsymbol{s}_j|B,\Gamma)}[\boldsymbol{s}_j] = \Gamma - \Gamma L^T \left(\Sigma_\epsilon + L\Gamma L^T\right)^{-1} L\Gamma, \quad \forall j, \tag{5}$$

where $\boldsymbol{s}_j$ denotes the $j$-th column of $S$ and individual columns are uncorrelated. However, since $\Gamma$ is actually not known, a suitable approximation $\hat{\Gamma} \approx \Gamma$ must first be found. One principled way to accomplish this is to integrate out the sources $S$ and then maximize

$$p(B|\Gamma) = \int p(B|S)p(S|\Gamma)dS \propto \exp\left(-\frac{1}{2}B^T \Sigma_b^{-1} B\right), \quad \Sigma_b \triangleq \Sigma_\epsilon + L\Gamma L^T. \tag{6}$$

This is equivalent to minimizing the cost function

$$\mathcal{L}(\Gamma) \triangleq -2\log p(B|\Gamma)p(\Gamma) \equiv \mathrm{trace}\left[C_b \Sigma_b^{-1}\right] + \log|\Sigma_b,|, \tag{7}$$

where $C_b \triangleq n^{-1} BB^T$ is the empirical covariance, and is sometimes referred to as type-II maximum likelihood, evidence maximization, or empirical Bayes [1].

The first term of (7) is a measure of the dissimilarity between the empirical data covariance $C_b$ and the model data covariance $\Sigma_b$; in general, this factor encourages $\Gamma$ to be large. The second term provides a regularizing or sparsifying effect, penalizing a measure of the volume formed by the model covariance $\Sigma_b$.[1] Since the volume of any high dimensional space is more effectively reduced by collapsing individual dimensions as close to zero as possible (as opposed to incrementally reducing all dimensions isometrically), this penalty term promotes a model covariance that is maximally degenerate (or non-spherical), which pushes elements of $\Gamma$ to exactly zero. This intuition is supported theoretically by the results in Section 4.

Given some type-II ML estimate $\hat{\Gamma}$, we obtain the attendant empirical prior $p(S|\hat{\Gamma})$. To the extent that this 'learned' prior is realistic, the resulting posterior $p(S|B, \hat{\Gamma})$ quantifies regions of significant current density and point estimates for the unknown source dipoles $S_i$ can be obtained by evaluating the posterior mean computed using (4). If a given $\hat{\Gamma}_i \to 0$ as described above, then the associated $\hat{S}_i$ computed using (4) also becomes zero. It is this pruning mechanism that naturally chooses the number of active dipoles.

## 3 Algorithm Derivation

Given $\Sigma_\epsilon$ and $\Gamma$, computing the posterior on $S$ is trivial. Consequently, determining these unknown quantities is the primary estimation task. We will first derive an algorithm for computing $\Gamma$ assuming $\Sigma_\epsilon$ is known. Later in Section 3.2, we will describe a powerful procedure for learning $\Sigma_\epsilon$.

### 3.1 Learning the Hyperparameters $\Gamma$

The primary objective of this section is to minimize (7) with respect to $\Gamma$. Of course one option is to treat the problem as a general nonlinear optimization task and perform gradient descent or some other generic procedure. Related methods in the MEG literature rely, either directly or indirectly, on a form of the EM algorithm [3, 8]. However, these algorithms are exceedingly slow when $d_s$ is large and they have not been extended to handle flexible orientations. Consequently, here we derive an alternative optimization procedures that expands upon ideas from [8, 12], handles arbitrary/unknown dipole orientations, and converges quickly.

To begin, we note that $\mathcal{L}(\Gamma)$ only depends on the data $B$ through the $d_b \times d_b$ sample correlation matrix $C_b$. Therefore, to reduce the computational burden, we replace $B$ with a matrix $\widetilde{B} \in \mathbb{R}^{d_b \times \mathrm{rank}(B)}$ such that $\widetilde{B}\widetilde{B}^T = C_b$. This removes any per-iteration dependency on $d_t$, which can potentially be large, without altering that actual cost function. It also implies that, for purposes of computing $\Gamma$, the number of columns of $S$ is reduced to match $\mathrm{rank}(B)$. We now re-express the cost function $\mathcal{L}(\Gamma)$ in an alternative form leading to convenient update rules and, by construction, a proof that $\mathcal{L}\left(\Gamma^{(k+1)}\right) \leq \mathcal{L}\left(\Gamma^{(k)}\right)$ at each iteration.

First, the data fit term can be expressed as

$$\text{trace}\left[C_b \Sigma_b^{-1}\right] = \min_X \left[\left\|\widetilde{B} - \sum_{i=1}^{d_s} L_i X_i\right\|_{\Sigma_\epsilon^{-1}}^2 + \sum_{i=1}^{d_s} \|X_i\|_{\Gamma_i^{-1}}^2\right], \tag{8}$$

where $X \triangleq \left[X_1^T, \ldots, X_{d_s}^T\right]^T$ is a matrix of auxiliary variables. Likewise, because the log-determinant term of $\mathcal{L}(\Gamma)$ is concave in $\Gamma$, it can be expressed as a minimum over upper-bounding hyperplanes via

$$\log|\Sigma_b| = \min_Z \left[\sum_{i=1}^{d_s} \text{trace}\left(Z_i^T \Gamma_i\right) - h^*(Z)\right], \tag{9}$$

where $Z \triangleq \left[Z_1^T, \ldots, Z_{d_s}^T\right]^T$ and $h^*(Z)$ is the concave conjugate of $\log|\Sigma_b|$. For our purposes below, we will never actually have to compute $h^*(Z)$. Dropping the minimizations and combining terms from (8) and (9) leads to the modified cost function

$$\mathcal{L}(\Gamma, X, Z) = \left\|\widetilde{B} - \sum_{i=1}^{d_s} \widetilde{L}_i X_i\right\|_{\Sigma_\epsilon^{-1}}^2 + \sum_{i=1}^{d_s} \left[\|X_i\|_{\Gamma_i^{-1}}^2 + \text{trace}\left(Z_i^T \Gamma_i\right)\right] - h^*(Z), \tag{10}$$

where by construction $\mathcal{L}(\Gamma) = \min_X \min_Z \mathcal{L}(\Gamma, X, Z)$. It is straightforward to show that if $\{\hat{\Gamma}, \hat{X}, \hat{Z}\}$ is a local (global) minimum to $\mathcal{L}(\Gamma, X, Z)$, then $\hat{\Gamma}$ is a local (global) minimum to $\mathcal{L}(\Gamma)$.

Since direct optimization of $\mathcal{L}(\Gamma)$ may be difficult, we can instead iteratively optimize $\mathcal{L}(\Gamma, X, Z)$ via coordinate descent over $\Gamma$, $X$, and $Z$. In each case, when two are held fixed, the third can be globally minimized in closed form. This ensures that each cycle will reduce $\mathcal{L}(\Gamma, X, Z)$, but more importantly, will reduce $\mathcal{L}(\Gamma)$ (or leave it unchanged if a fixed-point or limit cycle is reached). The associated update rules from this process are as follows.

The optimal $X$ (with $\Gamma$ and $Z$ fixed) is just the standard weighted minimum-norm solution given by

$$X_i^{\text{new}} \to \Gamma_i L_i^T \Sigma_b^{-1} \widetilde{B} \tag{11}$$

for each $i$. The minimizing $Z$ equals the slope at the current $\Gamma$ of $\log|\Sigma_b|$. As such, we have

$$Z_i^{\text{new}} \to \nabla_{\Gamma_i} \log|\Sigma_b| = L_i^T \Sigma_b^{-1} L_i. \tag{12}$$

With $Z$ and $X$ fixed, computing the minimizing $\Gamma$ is a bit more difficult because of the constraint $\Gamma_i \in H^+$ for all $i$, where $H^+$ is the set of positive-semidefinite, symmetric $d_c \times d_c$ covariance matrices. To obtain each $\Gamma_i$, we must solve

$$\Gamma_i^{\text{new}} \to \arg \min_{\Gamma_i \in H^+} \left[\|X_i\|_{\Gamma_i^{-1}}^2 + \text{trace}\left(Z_i^T \Gamma_i\right)\right] \tag{13}$$

An unconstrained solution will satisfy

$$\nabla_{\Gamma_i} \mathcal{L}(\Gamma_i, X_i, Z_i) = 0, \tag{14}$$

which, after computing the necessary derivatives and re-arranging terms gives the equivalent condition

$$X_i X_i^T = \Gamma_i Z_i \Gamma_i. \tag{15}$$

There are multiple (unconstrained) solutions to this equation; we will choose the unique one that satisfies the constraint $\Gamma_i \in H^+$. This can be found using

$$\begin{aligned} X_i X_i^T &= Z_i^{-1/2} \left(Z_i^{1/2} X_i X_i^T Z_i^{1/2}\right) Z_i^{-1/2} \\ &= Z_i^{-1/2} \left(Z_i^{1/2} X_i X_i^T Z_i^{1/2}\right)^{1/2} \left(Z_i^{1/2} X_i X_i^T Z_i^{1/2}\right)^{1/2} Z_i^{-1/2} \\ &= \left[Z_i^{-1/2} \left(Z_i^{1/2} X_i X_i^T Z_i^{1/2}\right)^{1/2} Z_i^{-1/2}\right] Z_i \left[Z_i^{-1/2} \left(Z_i^{1/2} X_i X_i^T Z_i^{1/2}\right)^{1/2} Z_i^{-1/2}\right]. \end{aligned} \tag{16}$$

This indicates the solution (or update equation)

$$\Gamma_i^{\text{new}} \to Z_i^{-1/2} \left(Z_i^{1/2} X_i X_i^T Z_i^{1/2}\right)^{1/2} Z_i^{-1/2}, \tag{17}$$

which is satisfies the constraint. And since we are minimizing a convex function of $\Gamma_i$ (over the constraint set), we know that this is indeed a minimizing solution.

In summary then, to estimate $\Gamma$, we need simply iterate (11), (12), and (17), and with each pass we are guaranteed to reduce (or leave unchanged) $\mathcal{L}(\Gamma)$. The per-iteration cost is linear in the number of voxels $d_s$ so the computational cost is relatively modest (it is quadratic in $d_b$, and cubic in $d_c$, but these quantities are relatively small). The convergence rate is orders of magnitude faster than EM-based algorithms such as those in [3, 8] (see Figure 1 (*right*) ).

## 3.2 Learning the Interference $\Sigma_\epsilon$

The learning procedure described in the previous section boils down to fitting a structured maximum likelihood covariance estimate $\Sigma_b = \Sigma_\epsilon + F\Gamma F^T$ to the data covariance $C_b$. The idea here is that $F\Gamma F^T$ will reflect the brain signals of interest while $\Sigma_\epsilon$ will capture all interfering factors, e.g., spontaneous brain activity, sensor noise, muscle artifacts, etc. Since $\Sigma_\epsilon$ is unknown, it must somehow be estimated or otherwise accounted for. Given access to pre-stimulus data (i.e., data assumed to have no signal/sources of interest), stimulus evoked factor analysis (SEFA) provides a powerful means of decomposing a data covariance matrix $C_b$ into signal and interference components. While details can be found in [4], SEFA computes the approximation

$$C_b \approx \Lambda + EE^T + AA^T, \tag{18}$$

where $E$ represents a matrix of learned interference factors, $\Lambda$ is a diagonal noise matrix, and $A$ is a matrix of signal factors. There are two ways to utilize this decomposition (more details can be found in [11]). First, we can simply set $\Sigma_\epsilon \to \Lambda + EE^T$ and proceed as in Section 3.1. Alternatively, we can set $\Sigma_\epsilon \to 0$ and then substitute $AA^T$ for $C_b$, i.e., run the same algorithm on a de-noised signal covariance. For technical reasons beyond the scope of this paper, it appears that algorithm performance may be superior when the latter paradigm is adopted.

# 4 Analysis of Theoretical Localization/Orientation Bias

Theoretical support for the proposed algorithm is possible in the context of estimation bias assuming simplified source configurations. For example, substantial import has been devoted to quantifying localization bias when estimating a single dipolar source. Recently it has been shown, both empirically and theoretically [5, 9], that the MVAB and sLORETA algorithms have zero location bias under this condition at high SNR. This has been extended to include certain empirical Bayesian methods [8, 12]. However, these results assume a single dipole with fixed, known orientation (or alternatively, that $d_c = 1$), and therefore do not formally handle source correlations or multi-component dipoles. The methods from [6, 13] also purport to address these issues, but no formal analyses are presented.

In contrast, despite being a complex, non-convex function, we now demonstrate that $\mathcal{L}(\Gamma)$ has very attractive bias properties regarding both localization and orientation. We will assume that the full lead-field $L \triangleq \left[L_1^T, \ldots, L_{d_s}^T\right]^T$ represents a sufficiently high sampling of the source space such that any active dipole component aligns with some lead-field columns. Unbiasedness can also be shown in the continuous case, but the discrete scenario is more straightforward and of course more relevant to any practical task.

Some preliminary definitions are required to proceed. We define the *empirical intra-dipole correlation matrix* at the $i$-th voxel as $C_{ii} \triangleq \frac{1}{d_t} S_i^T S_i$; non-zero off-diagonal elements imply that correlations are present. Except in highly contrived situations, this type of correlation will always exist. The *empirical inter-dipole correlation matrix* between voxels $i$ and $j$ is $C_{ij} \triangleq \frac{1}{d_t} S_i^T S_j$; any non-zero element implies the existence of a correlation. In practice, this form of correlation may or may not be present. With regard to the lead-field $L$, *spark* is defined as the smallest number of linearly dependent columns [2]. By definition then, $2 \leq \text{spark}(L) \leq d_b + 1$. Finally, $d_a$ denotes the number of active sources, i.e., the number of voxels whereby $\|S_i\| > 0$.

**Theorem 1.** In the limit as $\Sigma_\epsilon \to 0$ (high SNR) and assuming $d_a d_c < \text{spark}(L) - 1$, the cost function $\mathcal{L}(\Gamma)$ maintains the following two properties:

    1. For arbitrary $C_{ii}$ and $C_{ij}$, the unique global minimum $\Gamma^*$ produces a source estimate $S^* = \mathrm{E}_{p(S|B,\Gamma^*)}[S]$ computed using (4) that equals the generating source matrix $S$, i.e., it is

unbiased in both location and orientation for all active dipoles and correctly zeros out the inactive ones.

2. If $C_{ij} = 0$ for all active dipoles (although $C_{ii}$ is still arbitrary), then there are no local minima, i.e., the cost function is unimodal.

The proof has been deferred to [11]. In words, this theorem says that intra-dipole correlations do not disrupt the estimation process by creating local minima, and that the global minimum is always unbiased. In contrast, inter-dipole correlations can potentially create local minima, but they do not affect the global minimum. Empirically, we will demonstrate that the algorithm derived in Section 3 is effective at avoiding these local minima (see Section 5). With added assumptions these results can be extended somewhat to handle the inclusion of noise.

The cost functions from [8, 12] bear the closest resemblance to $\mathcal{L}(\Gamma)$; however, neither possesses the second attribute from Theorem 1. This is a very significant failing because, as mentioned previously, intra-dipole correlations are always present in each active dipole. Consequently, localization and orientation bias can occur because of convergence to a local minimum. The iterative Bayesian scheme from [13], while very different in structure, also directly attempts to estimate flexible orientations and handle, to some extent, source correlations. While details are omitted for brevity, we can prove that the full model upon which this algorithm is based fails to satisfy the first property of the theorem, so the corresponding global minimum can be biased. In contrast, beamformers and sLORETA are basically linear methods with no issue of global or local minima. However, the popular sLORETA and MVAB solutions will in general display a bias for multi-component dipoles ($d_c > 1$) or when multiple dipoles ($d_a > 1$) are present, regardless of correlations.

## 5 Empirical Evaluation

In this section we test the performance of our algorithm on both simulated and real data sets. We focus here on localization accuracy assuming strong source correlations and unknown orientations. While orientation estimates themselves are not shown for space considerations, accurate localization implicitly indicates that this confound has been adequately handled. More comprehensive experiments, including comparisons with additional algorithms, are forthcoming [11].

*Simulated Data*: We first conducted tests using simulated data with realistic source configurations. The brain volume was segmented into 5mm voxels and a two orientation ($d_c = 2$) forward leadfield was calculated using a spherical-shell model [7]. The data time course was partitioned into pre- and post-stimulus periods. In the pre-stimulus period (263 samples) there is only noise and interfering brain activity, while in the post-stimulus period (437 samples) there is the same (statistically) noise and interference factors plus source activity of interest. We used two noise conditions - Gaussian-noise and real-brain noise. In the former case, we seeded voxels with Gaussian noise in each orientation and then projected the activity to the sensors using the leadfield, producing colored Gaussian noise at the sensors. To this activity, we added additional Gaussian sensor noise. For the real-brain noise case, we used resting-state data collected from a human subject that is presumed to have on-going and spontaneous activity and sensor noise. In both the Gaussian and real-brain noise cases, the pre-stimulus activity was on-going and continued into the post-stimulus period, where the simulated source signals were added. Sources were seeded at locations in the brain as damped-sinusoids and this voxel activity was projected to the sensors. We could adjust both the signal-to-noise-plus-interefence ratio (SNIR) and the correlations between the different voxel time-courses to examine the algorithm performance on correlated sources and unknown dipole orientations.

We ran 100 simulations of three randomly seeded sources at different SNIR levels (-5, 0, 5, 10dB). The sources in these simulations always had an inter-dipole correlation coefficient of 0.5; intra-dipole correlations were present as well. We ran the simulation with both Gaussian-noise and real brain noise using a MVAB and our proposed method. In order to evaluate performance, we used the following test for a hit or miss. We drew spheres around each seeded source location and obtained the maximum voxel value in each sphere. Then we calculated the maximum voxel activation outside the three spheres. If the maximum inside each sphere was greater than the maximum outside all of the spheres, it was counted as a hit (in this way, we are implicitly accounting somewhat for false alarms). Each simulation could get a score or 0, 1 ,2 , or 3, with 3 being the best. Figure 1 (*left*) displays comparative results averaged over 100 trials with standard errors. Our method quite significantly outperforms the MVAB, which is designed to handle unknown orientations but has difficulty with

source correlations. Figure 1 (*middle*) shows a sample reconstruction on a much more complex source configuration composed of 10 dipolar sources. Finally, Figure 1 (*right*) gives an example of the relative convergence improvement afforded by our method relative to an EM implementation analogous to [3, 8]. We also wanted to test the performance on perfectly correlated sources with unknown orientations and compare it to other state-of-the-art Bayesian methods. An example using three such sources and 5 dB SNIR is given in Figure 2.

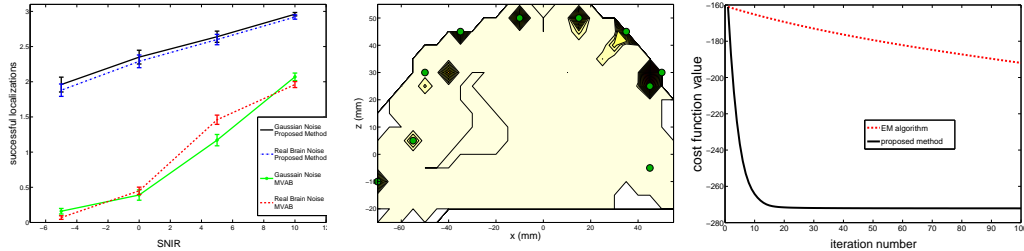

Figure 1: *Left*: Aggregate localization results for MVAB and the proposed method recovering three correlated sources with unknown orientations. *Middle*: Example reconstruction of 10 relatively shallow sources (green circles) using proposed method (MVAB performs poorly on this task). *Right*: Convergence rate of proposed method relative to a conventional EM implementation based on [3, 8].

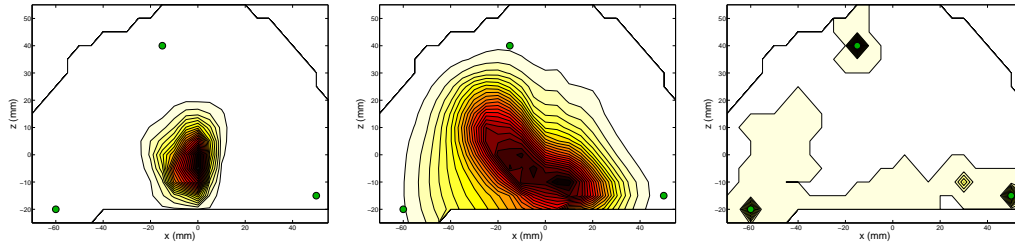

Figure 2: Reconstructions of three perfectly correlated dipoles (green circles) with unknown orientations using, *Left*: MVAB, *Middle*: variational Bayesian method from [13], *Right*: proposed method.

*Real Data*: Two stimulus-evoked data sets were collected from normal, healthy research subjects on a 275-channel CTF System MEG device. The first data set was a sensory evoked field (SEF) paradigm, where the subject's right index finger was tapped for a total of 256 trials. A peak is typically seen 50ms after stimulation in the contralateral (in this case, the left) somatosensory cortical area for the hand, i.e., dorsal region of the postcentral gyrus. The proposed algorithm was able to localize this activation to the correct area of somatosensory cortex as seen in Figure 3 (*left*) and the estimated time course shows the typical 50ms peak (data not shown). The second data set analyzed was an auditory evoked field (AEF) paradigm. In this paradigm the subject is presented tones binaurally for a total of 120 trials. There are two typical peaks seen after the presentation of an auditory stimulus, one at 50ms and one at 100ms, called the M50 and M100 respectively. The auditory processing of tones is bilateral at early auditory areas and the activations are correlated. The algorithm was able to localize activity in both primary auditory cortices and the time courses for these two activations reveal the M50 and M100. Figure 3 (*middle*) and (*right*) displays these results. The analysis of simple auditory paradigms is problematic because many source localization algorithms, such as the MVAB, do not handle the bilateral correlated sources well. We also ran MVAB on the AEF data and it localized activity to the center of the head between the two auditory cortices (data not shown).

## 6   Discussion

This paper derives a novel empirical Bayesian algorithm for MEG source reconstruction that readily handles multiple correlated sources with unknown orientations, a situation that commonly arises even with simple imaging tasks. Based on a principled cost function and fast, convergent update

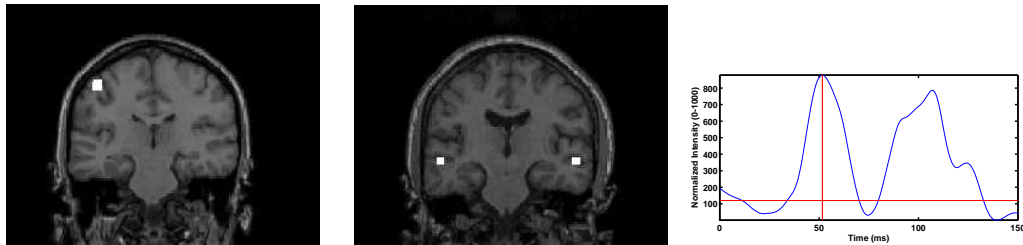

Figure 3: Real-world example *Left*: Somatosensory reconstruction. *Middle*: Bilateral auditory reconstruction. *Right*: Recovered timecourse from left auditory cortex (right auditory cortex, not shown, is similar).

rules, this procedure displays significant theoretical and empirical advantages over many existing methods. We have restricted most of our exposition and analyses to MEG; however, preliminary work with EEG is also promising. For example, on a real-world passive visual task where subjects viewed flashing foreground/background textured images, our method correctly localizes activity to the lateral occipital cortex while two state-of-the-art beamformers fail. This remains an active area of research.

## Footnotes

[1]The determinant of a matrix is equal to the product of its eigenvalues, a well-known volumetric measure.

# References

[1] J.O. Berger, *Statistical Decision Theory and Bayesian Analysis*, Springer-Verlag, New York, 2nd edition, 1985.

[2] D.L. Donoho and M. Elad, "Optimally sparse representation in general (nonorthogonal) dictionaries via $\ell_1$ minimization," *Proc. National Academy of Sciences*, vol. 100, no. 5, pp. 2197–2202, March 2003.

[3] K. Friston, L. Harrison, J. Daunizeau, S. Kiebel, C. Phillips, N. Trujillo-Barreto, R. Henson, G. Flandin, and J. Mattout, "Multiple sparse priors for the MEG/EEG inverse problem," *NeuroImage*, 2008 (in press).

[4] S.S. Nagarajan, H.T. Attias, K.E. Hild K.E., K. Sekihara, "A probabilistic algorithm for robust interference suppression in bioelectromagnetic sensor data," Stat Med. vol. 26, no. 21, pp. 3886–910 Sept. 2007.

[5] R.D. Pascual-Marqui, "Standardized low resolution brain electromagnetic tomography (sloreta): Technical details," *Methods and Findings in Experimental and Clinical Pharmacology*, vol. 24, no. Suppl D, pp. 5–12, 2002.

[6] M. Sahani and S.S. Nagarajan, "Reconstructing MEG sources with unknown correlations," *Advances in Neural Information Processing Systems 16*, 2004.

[7] J. Sarvas, "Basic mathematical and electromagnetic concepts of the biomagnetic inverse problem," *Phys. Med. Biol.*, vol. 32, pp. 11–22, 1987.

[8] M. Sato, T. Yoshioka, S. Kajihara, K. Toyama, N. Goda, K. Doya, and M. Kawato, "Hierarchical Bayesian estimation for MEG inverse problem," *NeuroImage*, vol. 23, pp. 806–826, 2004.

[9] K. Sekihara, M. Sahani, and S.S. Nagarajan, "Localization bias and spatial resolution of adaptive and non-adaptive spatial filters for MEG source reconstruction," *NeuroImage*, vol. 25, pp. 1056–1067, 2005.

[10] K. Uutela, M. Hamalainen, and E. Somersalo, "Visualization of magnetoencephalographic data using minimum current estimates," *NeuroImage*, vol. 10, pp. 173–180, 1999.

[11] D.P. Wipf, J.P. Owen, H.T. Attias, K. Sekihara, and S.S. Nagarajan "Robust Bayesian Estimation of the Location, Orientation, and Timecourse of Mutliple Correlated Neural Sources using MEG," *submitted*, 2009.

[12] D.P. Wipf, R.R. Ramírez, J.A. Palmer, S. Makeig, and B.D. Rao, "Analysis of empirical Bayesian methods for neuroelectromagnetic source localization," *Advances in Neural Information Processing Systems 19*, 2007.

[13] J.M. Zumer, H.T. Attias, K. Sekihara, and S.S. Nagarajan, "A probabilistic algorithm for interference suppression and source reconstruction from MEG/EEG data," *Advances in Neural Information Processing System 19*, 2007.
